# Label Selection on Graphs

**Andrew Guillory**
Department of Computer Science
University of Washington
guillory@cs.washington.edu

**Jeff Bilmes**
Department of Electrical Engineering
University of Washington
bilmes@ee.washington.edu

## Abstract

We investigate methods for selecting sets of labeled vertices for use in predicting the labels of vertices on a graph. We specifically study methods which choose a single batch of labeled vertices (i.e. offline, non sequential methods). In this setting, we find common graph smoothness assumptions directly motivate simple label selection methods with interesting theoretical guarantees. These methods bound prediction error in terms of the smoothness of the true labels with respect to the graph. Some of these bounds give new motivations for previously proposed algorithms, and some suggest new algorithms which we evaluate. We show improved performance over baseline methods on several real world data sets.

## 1 Introduction

In this work we consider learning on a graph. Assume we have an undirected graph of $n$ nodes given by a symmetric weight matrix $W$. The $i$th node in the graph has a label $y_i \in \{0, 1\}$ stored in a vector of labels $y \in \{0, 1\}^n$. We want to predict all of $y$ from the labels $y_L$ for a labeled subset $L \subset V = [n]$. $V$ is the set of all vertices. We use $\hat{y} \in \{0, 1\}^n$ to denote our predicted labels. The number of incorrect predictions is $||y - \hat{y}||^2$.

Graph-based learning is an interesting alternative to traditional feature-based learning. In many problems, graph representations are more natural than feature vector representations. When classifying web pages, for example, edge weights in the graph may incorporate information about hyperlinks. Even when the original data is represented as feature vectors, transforming the data into a graph (for example using a Gaussian kernel to compute weights between points) can be convenient for exploiting properties of a data set.

In order to bound prediction error, we assume that the labels are smoothly varying with respect to the underlying graph. The simple smoothness assumption we use is that $\sum_{i,j} W_{i,j} |y_i - y_j|$ is small. Here $||$ denotes absolute value, but the labels are binary so we can equivalently use squared difference. This smoothness assumption has been used by graph-based semi-supervised learning algorithms which compute $\hat{y}$ using a labeled set $L$ chosen uniformly at random from $V$ [Blum and Chawla, 2001, Hanneke, 2006, Pelckmans et al., 2007, Bengio et al., 2006] and by online graph labeling methods that operate on an adversarially ordered stream of vertices [Pelckmans and Suykens, 2008, Brautbar, 2009, Herbster et al., 2008, 2005, Herbster, 2008]

In this work we consider methods that make use of the smoothness assumption and structure of the graph in order to both select $L$ as well as make predictions. Our hope is to achieve higher prediction accuracy as compared to random label selection and other methods for choosing $L$. We are particularly interested in batch offline methods which select $L$ up front, receive $y_L$ and then predict $\hat{y}$. The single batch, offline label selection problem is important in many real-world applications because it is often the case that problem constraints make requesting more than one batch of labels very costly. For example, if requesting a label involves a time consuming, expensive experiment (potentially

involving human subjects), it may be significantly less costly to run a single batch of experiments in parallel as compared to running experiments in series.

We give several methods which, under the assumption $\sum_{i,j} W_{i,j}|y_i - y_j|$ is small, guarantee the prediction error $||y - \hat{y}||^2$ will also be small. Some of the bounds provide interesting justifications for previously used methods, and we show improved performance over random label selection and baseline submodular maximization methods on several real world data sets.

## 2 General Worst Case Bound

We first give a simple worst case bound on prediction error in terms of label smoothness using few assumptions about the method used to select labels or make predictions. In fact, the only assumption we make is that the predictions are consistent with the set of labeled points (i.e. $\hat{y}_L = y_L$). The bound motivates an interesting method for selecting labeled points and provides a new motivation for a standard prediction method Blum and Chawla [2001] when used with arbitrarily selected $L$. The bound also forms the basis of the other bounds we derive which make additional assumptions.

Define the graph cut function $\Gamma(A, B) \triangleq \sum_{i \in A, j \in B} W_{i,j}$. Let

$$\Psi(L) \triangleq \min_{T \subseteq (V \setminus L) \neq 0} \frac{\Gamma(T, (V \setminus T))}{|T|}$$

Note this function is different from normalized cut (also called sparsest cut). In this function, the denominator is simply $|T|$ while for normalized cut the denominator is $\min(|T|, |V \setminus T|)$. This difference is important: computing normalized cut is NP-hard, but we will show $\Psi(L)$ can be computed in polynomial time. $\Psi(L)$ measures how easily we can cut a large portion of the graph away from $L$. If $\Psi(L)$ is small, then we can separate many nodes from $L$ without cutting very many edges. We show that $\Psi(L)$ where $L$ is the set of labeled vertices measures to what extent prediction error can be high relative to label smoothness. This makes intuitive sense because if $\Psi(L)$ is small than there is a large set of unlabeled nodes which are weakly connected to the remainder of the graph (including $L$).

**Theorem 1.** *For any $\hat{y}$ consistent with a labeled set $L$*

$$||y - \hat{y}||^2 \leq \frac{1}{2\Psi(L)} \sum_{i,j} W_{i,j}(|y_i - y_j| \oplus |\hat{y}_i - \hat{y}_j|) \leq \frac{1}{2\Psi(L)} (\sum_{i,j} W_{i,j}|y_i - y_j| + \sum_{i,j} W_{i,j}|\hat{y}_i - \hat{y}_j|)$$

*where $\oplus$ is the XOR operator.*

*Proof.* Let $I$ be the set of incorrectly classified points. First note that $I \cap L = \emptyset$ (none of the labeled points are incorrectly classified).

$$|I| = \Gamma(I, V \setminus I) \frac{|I|}{\Gamma(I, V \setminus I)} \leq \frac{\Gamma(I, V \setminus I)}{\Psi(L)}$$

Note that for all of the edges $(i, j)$ counted in $\Gamma(I, V \setminus I)$, $\hat{y}_i = \hat{y}_j$ implies $y_i \neq y_j$ and $\hat{y}_i \neq \hat{y}_j$ implies $y_i = y_j$. Then

$$|I| \leq \frac{1}{2\Psi(L)} \sum_{i,j} W_{i,j}(|y_i - y_j| \oplus |\hat{y}_i - \hat{y}_j|)$$

The $\frac{1}{2}$ term is introduced because the sum double counts edges.  $\square$

This bound is tight when the set of incorrectly classified points $I$ is one of the sets minimizing $\min_{T \subseteq (V \setminus L) \neq 0} \Gamma(T, (V \setminus T))/|T|$.

This bound provides an interesting justification for the algorithm in Blum and Chawla [2001] and related methods when used with arbitrarily selected labeled sets. The term involving the predicted labels, $\sum_{i,j} W_{i,j}|\hat{y}_i - \hat{y}_j|$, is the objective function minimized under the constraint $\hat{y}_L = y_L$ by the algorithm of Blum and Chawla [2001]. When this is used to compute $\hat{y}$, the bound simplifies.

```
ComputeCut(L)                            MaximizeΨ(L, k)
   T' ← V \ L                               L ← ∅
   repeat                                    repeat
      T ← T'                                    T ← ComputeCut(L)
      λ ← Γ(T,V\T)/|T|                          i ← random vertex in T
      T' ← argmin Γ(A, V \ A) − λ|A|            L ← L ∪ {i}
            A⊆(V\L)                           until |L| = k
   until Γ(T', V \ T') − λ|T'| = 0            return L
   return T
```

Figure 1: Left: Algorithm for computing $\Psi(L)$. Right: Heuristic for maximizing $\Psi(L)$.

**Lemma 1.** *If*

$$\hat{y} = \mathrm{argmin}_{\hat{y}\in\{0,1\}^n:\hat{y}_L=y_L} \sum_{i,j} W_{i,j}|\hat{y}_i - \hat{y}_j|$$

*for a labeled set L then*

$$||y - \hat{y}||^2 \leq \frac{1}{\Psi(L)}\sum_{i,j} W_{i,j}|y_i - y_j|$$

*Proof.* When we choose $\hat{y}$ in this way $\sum_{i,j} W_{i,j}|\hat{y}_i - \hat{y}_j| \leq \sum_{i,j} W_{i,j}|y_i - y_j|$ and the lemma follows from Theorem 1. ☐

Label propagation solves a version of this problem in which $\hat{y}$ is real valued [Bengio et al., 2006].

The bound also motivates a simple label selection method. In particular, we would like to select a labeled $L$ set that maximizes $\Psi(L)$. We first describe how to compute $\Psi(L)$ for a fixed $L$. Computing $\Psi(L)$ is related to computing

$$\min_{T\subseteq(V\setminus L)} \Gamma(T, V \setminus T) - \lambda|T| \tag{1}$$

with parameter $\lambda > 0$. The following result is paraphrased from Fujishige [2005] (pages 248-249).

**Theorem 2.** $\lambda' = \min_T \frac{f(T)}{g(T)}$ *if and only if*

$$\forall \lambda \leq \lambda' \quad \min_T f(T) - \lambda g(T) = 0$$

*and*

$$\forall \lambda > \lambda' \quad \min_T f(T) - \lambda g(T) < 0$$

We can compute Equation 1 for all $\lambda$ via a parametric maxflow/mincut computation (it is known there are no more than $n - 1$ distinct solutions). This gives a polynomial time algorithm for computing $\Psi(L)$. Note this theorem is for unconstrained minimization of $T$, but restricting $T \cap L = \emptyset$ does not change the result: this constraint simply removes elements from the ground set. In practice, this constraint can be enforced by contracting the graph used in the flow computations or by giving certain edges infinite capacity.

As an alternative to solving the parametric flow problem, we can find the desired $\lambda$ value through an iterative method [Cunningham, 1985]. The left of Figure 1 shows this approach. The algorithm takes in a set $L$ and computes $\mathrm{argmin}_{T\subseteq(V\setminus L)\neq0} \Gamma(T, (V\setminus T))/|T|$. The correctness proof is simple. When the algorithm terminates, we know $\lambda \geq \lambda' = \min_{T\subseteq(V\setminus L)\neq0} \Gamma(T, (V \setminus T))/|T|$ because we set $\lambda$ to be $\Gamma(T, (V \setminus T))/|T|$ for a particular $T$. By Theorem 2 and the termination condition, we also know $\lambda \leq \lambda'$ and can conclude $\lambda = \lambda'$ and the set $T$ returned achieves this minimum. One can also show the algorithm terminates in at most $|V|$ iterations [Cunningham, 1985].

Having shown how to compute $\Psi(L)$, we now consider methods for maximizing it. $\Psi$ is neither submodular nor supermodular. This seems to rule out straightforward set function optimization. In our experiments, we try a simple heuristic based on the following observation: for any $L$, if $\Psi(L') > \Psi(L)$ then it must be the case that $L'$ intersects one of the cuts minimizing $\min_{T\subseteq(V\setminus L)\neq\emptyset} \Gamma(T, (V \setminus$

$T))/|T|$. In other words, in order to increase $\Psi(L)$ we must necessarily include a point from the current cut. Our heuristic is then to simply add a random element from this cut to $L$. The right of Figure 1 shows this method.

Several issues remain. First, although we have proposed a reasonable heuristic for maximizing $\Psi(L)$, we do not have methods for maximizing it exactly or with guaranteed approximation. Aside from knowing the function is not submodular or supermodular, we also do not know the hardness of the problem. In the next section, we describe a lower bound on the $\Psi$ function based on a notion of graph covering. This lower bound can be maximized approximately via a simple algorithm and has a well understood hardness of approximation. Second, we have found in experimenting with our heuristic for maximizing $\Psi(L)$ that the function can be prone to imbalanced cuts; the computed cuts sometimes contain all or most of the unselected points $V \setminus L$ and other times focus on small sets of outliers. We give a third bound on error which attempts to address some of this sensitivity.

## 3   Graph Covering Algorithm

The method we consider in this section uses a notion of graph covering. We say a set $L$ $\alpha$-covers the graph if $\forall i \in V$ either $i \in L$ or $\sum_{j \in L} W_{i,j} \geq \alpha$. In other words, every node in the graph is either in $L$ or connected with total weight at least $\alpha$ to nodes in $L$ (or both). This is a simple real valued extension of dominating sets. A dominating set is a set $L \subseteq V$ such that $\forall i \in V$ either $i \in L$ or a neighbor of $i$ is in $L$ (or both). This notion of covering is related to the $\Psi$ function discussed in the previous section. In particular, if a set $L$ $\alpha$-covers a graph than it is necessarily the case that $\Psi(L) \geq \alpha$. The converse does not hold, however. In other words, $\alpha$ is a lower bound on $\Psi(L)$. Then, $\alpha$ can replace $\Psi(L)$ in the bound in the previous section for a looser upper bound on prediction error. Although the bound is looser, compared to maximizing $\Psi(L)$ we better understand the complexity of computing an $\alpha$-cover.

**Corollary 1.** *For any $\hat{y}$ consistent with a labeled set $L$ that is an $\alpha$-cover*

$$||y - \hat{y}||^2 \leq \frac{1}{2\alpha} \sum_{i,j} W_{i,j}(|y_i - y_j| \oplus |\hat{y}_i - \hat{y}_j|) \leq \frac{1}{2\alpha} \left( \sum_{i,j} W_{i,j}|y_i - y_j| + \sum_{i,j} W_{i,j}|\hat{y}_i - \hat{y}_j| \right)$$

*where $\oplus$ is the XOR operator.*

Similar to Lemma 1, by making additional assumptions concerning the prediction method used we can derive a slightly simpler bound. In particular, for a labeled set $L$ that is an $\alpha$ cover, we assume unlabeled nodes are labeled with the weighted majority vote of neighbors in $L$. In other words, set $\hat{y}_i = y_i$ for $i \in L$, and set $\hat{y}_i = y'$ for $i \notin L$ with $y'$ such that $\sum_{j \in L : y_j = y'} W_{i,j} \geq \sum_{j \in L : y_j \neq y'} W_{i,j}$. With this prediction method we get the following bound.

**Lemma 2.** *If $L$ is an $\alpha$-cover and $V \setminus L$ is labeled according to majority vote*

$$||y - \hat{y}||^2 \leq \frac{1}{\alpha} \sum_{i,j} W_{i,j}|y_i - y_j|(1 - |\hat{y}_i - \hat{y}_j|) \leq \frac{1}{\alpha} \sum_{i,j} W_{i,j}|y_i - y_j|$$

*Proof.* The right hand side follows immediately from the middle expression, so we focus on the first inequality. For every incorrectly labeled node, there is a set of nodes $L_i = \{j \in L : \hat{y}_i = \hat{y}_j\}$ which satisfies $y_i \neq y_j \forall j \in L_i$, and $\sum_{j \in L_i} W_{i,j} \geq \alpha/2$. We then have for every incorrectly labeled node a unique set of edges with total weight at least $\alpha/2$ included inside the summation in the middle expression. □

In computing an $\alpha$-cover, we want to solve.

$$\min_{L \subseteq V} |L| : F(L) \geq \alpha$$

Where

$$F(L) \triangleq \min_{i \in V \setminus L} \sum_{j \in L} W_{i,j} = F'(L) \triangleq \min_{i \in V} \sum_{j \in L} W'_{i,j}$$

where $W'_{i,j} = W_{i,j}$ for $i \neq j$ and $W'_{i,i} = \infty$. $F'$ is the minimum of a set of modular functions. $F'$ is neither supermodular nor submodular. However, we can still compute an approximately minimal

$\alpha$-cover using a trick introduced by Krause et al. [2008]. In particular, Krause et al. [2008] point out that

$$\min_{i \in V} \sum_{j \in L} W'_{i,j} \geq \alpha \Leftrightarrow \sum_i \frac{1}{n} \min(\sum_{j \in L} W'_{i,j}, \alpha) \geq \alpha$$

Also, $\min(\sum_{j \in L} W'_{i,j}, \alpha)$ is submodular, and the sum of submodular functions is submodular. Then, we can replace $F'$ with

$$F'_\alpha(L) = \sum_i \frac{1}{n} \min(\sum_{j \in L} W'_{i,j}, \alpha)$$

and solve

$$\min_{L \subseteq V} |L| : F'_\alpha(L) \geq \alpha$$

This is a submodular set cover problem. The greedy algorithm has approximation guarantees for this problem for integer valued functions [Krause et al., 2008]. For binary weight graphs the approximation is $O(\log n)$. For real valued functions, it's possible to round the function values to get an approximation guarantee. In practice, we apply the greedy algorithm directly.

As previously mentioned, $\alpha$-covers can be seen as real valued generalizations of dominating sets. In particular, an $\alpha$-cover is a dominating set for binary weight graphs and $\alpha = 1$. The hardness of approximation results for finding a minimum size dominating set then carry over to the more general $\alpha$-cover problem. The next theorem shows that the $\alpha$-cover problem is NP-hard and in fact the greedy algorithm for computing an $\alpha$-cover is optimal up to constant factors for $\alpha = 1$ and binary weight graphs. It is based on the well known connection between finding a minimum dominating set problem and finding a minimum set cover.

**Theorem 3.** *Finding the smallest dominating set $L$ in a binary weight graph is $NP$-complete. Furthermore, if there is some $\epsilon > 0$ such that a polynomial time algorithm approximates the smallest dominating set within $(1 - \epsilon) \ln(n/2)$ then $NP \subset TIME(n^{O(\log \log n)})$.*

We have so far discussed computing a small $\alpha$ cover for a fixed $\alpha$. If we instead have a fixed label budget and want to maximize $\alpha$, we can do so by performing binary search over $\alpha$. This is the approach used by Krause et al. [2008] and gives a bi-approximation.

## 4  Normalized Cut Algorithm

In this section we consider an algorithm that clusters the data set and replaces the $\Psi$ function with a normalized cut value. The normalized cut value for a set $T \subset V$ is

$$\frac{\Gamma(T, V \setminus T)}{\min(|T|, |V \setminus T|)}$$

In other words, normalized cut is the ratio between the cut value for $T$ and minimum of the size of $T$ and its complement. Computing the minimum normalized cut for a graph is NP-hard.

Consider the following method: 1) partition the set of nodes $V$ into clusters $S_1, S_2, ...S_k$, 2) for each cluster request sufficient labels to estimate the majority class with probability at least $1 - \delta/k$, and 3) label all nodes in each cluster with the majority label for that cluster. Here the probability $1 - \delta/k$ is with respect to the choice of the labeled nodes used to estimate the majority class for each cluster.

**Theorem 4.** *Let $S_1, S_2, ...S_k$ be a partition of $V$, and assume we have estimates of the majority class of each $S_l$ each of which are accurate with probability at least $1 - \delta/k$. If $\hat{y}$ labels every $i \in S_l$ according to the estimated majority label for $S_l$ then with probability at least $1 - \delta$*

$$||y - \hat{y}||^2 \leq \sum_l \frac{1}{2\phi_l} \sum_{i,j \in S_l} W_{i,j}|y_i - y_j| \leq \frac{1}{2\phi} \sum_{i,j} W_{i,j}|y_i - y_j|$$

*where*

$$\phi_l = \min_{T \subset S_l} \frac{\Gamma(T, S_l \setminus T)}{\min(|T|, |S_l \setminus T|)}$$

*and*

$$\phi = \min_l \phi_l$$

*Proof.* By the union bound, the estimated majority labels for all of the clusters are correct with probability at least $1 - \delta$. Let $I$ be the set of incorrectly labeled nodes (errors). We consider the intersection of $I$ with each of the clusters. Let $I_l \triangleq |I \cap S_l|$. $I = \bigcup_{l=1}^{k} I_l$ Note that $|I_l| \leq |S_l \setminus I_l|$ since we labeled cluster according to the majority label for the cluster. Then

$$|I| = \sum_l |I_l| = \sum_l \Gamma(I_l, S_l \setminus I_l) \frac{|I_l|}{\Gamma(I_l, S_l \setminus I_l)}$$

$$= \sum_l \Gamma(I_l, S_l \setminus I_l) \frac{\min(|I_l|, |S_l \setminus I_l|)}{\Gamma(I_l, S_l \setminus I_l)}$$

$$\leq \sum_l \frac{\Gamma(I_l, S_l \setminus I_l)}{\phi_l}$$

For any $i, j$, with $i \in I_l$ and $j \in S_l \setminus I_l$, we must have $y_i \neq y_j$. Also, for any $i, j$ with $y_i \neq y_j$ and $i, j \in S_l$, either $i \in I_l$ or $j \in I_l$. In other words, there is a one-to-one correspondence between 1) edges $i, j$ for which $i, j \in S_l$ and either $i \in I_l$ or $j \in I_l$ and 2) edges $i, j$ for which $i, j \in S_l$ and $y_i \neq y_j$. The desired result then follows. $\qquad\square$

Note in practice we only label the unlabeled nodes in each cluster using the majority label estimates. Using the true labels for the labeled nodes only decreases error, so the theorem still holds.

In this bound, $\phi$ is a measure of the density of the clusters. Computing $\phi_l$ for a particular cluster is NP-hard, but there are approximation algorithms. However, we are not aware of approximation algorithms for computing a partition such that $\phi$ is maximized. This is different from the standard normalized cut clustering problem; we do not care if clusters are strongly connected to each other only that each cluster is internally dense. In our experiments, we try several standard clustering algorithms and achieve good real world performance, but it remains an interesting open question to design a clustering algorithm for directly maximizing $\phi$. An approach we have not yet tried is to use the error bound to choose between the results of different clustering algorithms.

We now consider the problem of estimating the majority class for a cluster. If we uniformly sample labels from a cluster, standard results give that the probability of incorrectly estimating the majority decreases exponentially with the number of labels if the fraction of nodes in the minority class is bounded away from $1/2$ by a constant. We now show that if the labels are sufficiently smooth and the cluster is sufficiently dense then the fraction of nodes in the minority class is small.

**Theorem 5.** *The fraction of nodes in the minority class of $S$ is at most*

$$\frac{\sum_{i,j \in S} W_{i,j} |y_i - y_j|}{\phi |S|}$$

*where*

$$\phi = \min_{T \subset S} \frac{\Gamma(T, S \setminus T)}{\min(|T|, |S \setminus T|)}$$

*Proof.* Let $S^-$ be the set of nodes belonging to the minority class and $S^+$ be the set of nodes belonging to the other class. Let $f$ be the fraction of nodes in the minority class.

$$f = \frac{|S^-|}{|S|} = \frac{|S^-|}{|S|} \frac{\sum_{i,j \in S} W_{i,j} |y_i - y_j|}{\min(|S^+|, |S^-|)} \frac{\min(|S^+|, |S^-|)}{\sum_{i,j \in S} W_{i,j} |y_i - y_j|}$$

$$= \frac{\sum_{i,j \in S} W_{i,j} |y_i - y_j|}{|S|} \frac{\min(|S^+|, |S^-|)}{\Gamma(S^+, S^-)}$$

$$\leq \frac{\sum_{i,j \in S} W_{i,j} |y_i - y_j|}{\phi |S|}$$

$\qquad\square$

If we have an estimate of the smoothness of the labels in a cluster, we can use this bound and an approximation of $\phi$ to determine the number of labels needed to estimate the majority class with high confidence. In our experiments, we simply request a single label per cluster.

|          | Spectral       | $k$-Cut        | METIS          | $\Psi$         | Baseline        |
|----------|----------------|----------------|----------------|----------------|-----------------|
| Digit1/10 | 9.54 (4.42)   | 50.02 (1.04)   | **4.93 (4.05)** | 49.92 (3.18)  | 20.90 (15.67)   |
| Text/10  | 37.64 (8.64)   | 50.03 (0.3)    | **34.76 (6.05)** | 50.05 (0.06) | 45.91 (7.96)    |
| BCI/10   | 50.13 (2.16)   | 50.16 (0.64)   | **49.68 (2.63)** | 50.32 (0.55) | 50.12 (1.32)    |
| USPS/10  | 15.22 (6.22)   | 31.53 (23.65)  | **8.15 (5.51)** | 20.07 (2.70)  | 15.87 (4.82)    |
| g241c/10 | 39.63 (5.67)   | 50.03 (0.03)   | **29.18 (7.28)** | 50.29 (0.07) | 47.26 (5.19)    |
| g241d/10 | **22.31 (7.06)** | 50.02 (0.23) | 22.57 (7.26)   | 50.01 (0.09)  | 48.46 (3.39)    |
| Digit1/100 | 4.47 (1.35)  | 50.07 (1.46)   | 3.24 (0.76)    | 2.60 (0.83)   | **2.57 (0.67)** |
| Text/100 | 31.67 (2.41)   | 50.26 (2.73)   | 32.57 (1.88)   | 48.34 (0.67)  | **26.82 (3.88)** |
| BCI/100  | 47.37 (2.80)   | 50.14 (0.5)    | **45.35 (1.91)** | 48.17 (1.87) | 47.48 (2.99)    |
| USPS/100 | **6.23 (1.49)** | 31.13 (26.31) | 9.28 (1.38)    | 10.17 (0.39)  | 6.33 (2.46)     |
| g241c/100 | 44.31 (2.09)  | 50.02 (0.18)   | **37.47 (2.13)** | 52.48 (0.37) | 42.86 (4.50)    |
| g241d/100 | 41.70 (2.44)  | 50.03 (0.18)   | **35.96 (1.99)** | 50.33 (0.21) | 41.56 (4.34)    |

Table 1: Error rate mean (standard deviation) for different data set, label count, method combinations.

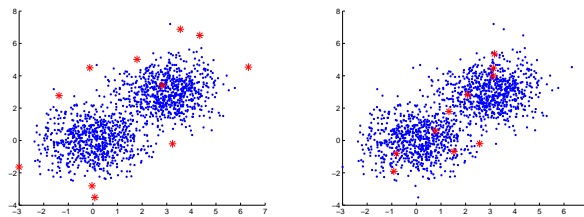

Figure 2: Left: Points selected by the $\Psi$ function maximization method. Right: Points selected by the spectral clustering method.

# 5 Experiments

We experimented with a method based on Lemma 1. We use the randomized method for maximizing $\Psi$ and then predict with min-cuts [Blum and Chawla, 2001]. We also tried a method based on Theorem 4. We cluster the data then label each cluster according to a single randomly chosen point. We chose the number of clusters to be equal to the number of labeled points observing that if a cluster is split evenly amongst the two classes then we will have a high error rate regardless of how well we estimate the majority class. We tried three clustering algorithms: a spectral clustering method [Ng et al., 2001], the METIS package for graph partitioning [Karypis and Kumar, 1999], and a $k$-cut approximation algorithm [Saran and Vazirani, 1995, Gusfield, 1990]. As a baseline we use random label selection and prediction using the label propagation method of Bengio et al. [2006] with $\epsilon = 10^{-6}$ and $\mu = 10^{-6}$ and class mass normalization. We also experimented with a method motivated by the graph covering bound, but for lack of space we omit these results.

We used six benchmark data sets [Chapelle et al., 2006]. We use graphs constructed with a Gaussian kernel with standard deviation chosen to be the average distance to the $k_1$th nearest neighbor divided by 3 (a similar heuristic is used by Chapelle et al. [2006]). We then make this graph sparse by removing the edge between node $i$ and $j$ unless $i$ is one of $j$'s $k_2$ nearest neighbors or $j$ is one of $i$'s $k_2$ nearest neighbors. We use 10 and 100 labels. We set $k_1$ and $k_2$ for each data set and label count to be the parameters which give the lowest average error rate for label propagation averaging over 100 trials and choosing from the set $\{5, 10, 50, 100\}$. We tune the graph construction parameters to give low error for the baseline method to ensure any bias is in favor of the baseline as opposed to the new methods we propose. We then report average error over 1000 trials in the 10 label case and 100 trials in the 100 label case for each combination of data set and algorithm.

Table 1 shows these results. We find that the $\Psi$ function method does not perform well. We found on most of the data sets the cuts found by the method included all or almost all of $V \setminus L$. In this case the points selected are essentially random. However, on the USPS data set and on some synthetic data sets we have tried, we have also observed the opposite behavior where the cuts are very small and seem to focus on small sets of outliers. Figure 2 shows an example of this. The $k$-cut method

also did not perform well. We've found this method has similar problems with outliers. We think these outlier sensitive methods are impractical for graphs constructed from real world data.

The results for the spectral clustering and METIS clustering methods, however, are quite encouraging. These methods performed well matching or beating the baseline method on the 10 label trials and in some cases significantly improving performance. The METIS method seems particularly robust. On the 100 label trials, performance was not as good. In general, we expect label selection to help more when learning from very few labels. The choice in clustering method seems to be of great practical importance. The clustering methods which work best seem to be methods which minimize normalize cut like objectives. This is not surprising given the presence of the normalized cut term in Theorem 4, but it is an open problem to give a clustering method for directly minimizing the bound.

We finally note that the numbers we report for our baseline method are in some cases significantly different than the published numbers [Chapelle et al., 2006]. This seems to be because of a variety of factors including differences in implementation as well as significant differences in experiment set up. We have also experimented with several heuristic modifications to our methods and compared our methods to simple greedy methods. One modification we tried is to use label propagation for prediction in conjunction with our label selection methods. We omit these results for lack of space.

# 6    Related Work

Previous work has also used clustering, covering, and other graph properties to guide label selection on graphs. We are, however, the first to our knowledge to give bounds which relate prediction error to label smoothness for single batch label selection methods. Most previous work on label selection methods for learning on graphs has considered active (i.e. sequential) label selection [Zhu and Lafferty, 2003, Pucci et al., 2007, Zhao et al., 2008, Wang et al., 2007, Afshani et al., 2007]. Afshani et al. [2007] show in this setting $O(c \log(n/c))$ where $c = \sum_{i,j} W_{i,j}|y_i - y_j|$ labels are sufficient and necessary to learn the labeling exactly under some balance assumptions. Without balance assumptions they show $O(c \log(1/\epsilon) + c \log(n/c))$ labels are sufficient to achieve an $\epsilon$ error rate. In some cases, our bounds are better despite considering only non sequential label selection. Consider the case where $c$ grows linearly with $n$ so $c/n = a$ for some constant $a > 0$. In this case, with the bound of Afshani et al. [2007] the number of labels required to achieve a fixed error rate $\epsilon$ also grows linearly with $n$. In comparison, our graph covering bound needs an $\alpha$-cover with $\alpha = a/\epsilon$. For some graph topologies, the size of such a cover can grow sublinearly with $n$ (for example if the graph contains large, dense clusters). Afshani et al. [2007] also use a kind of dominating set in their method, and it could be interesting to see if portions of their analysis could be adapted to the offline setting. Zhao et al. [2008] also use a clustering algorithm to select initial labels.

Other work has given generalization error bounds in terms of label smoothness [Pelckmans et al., 2007, Hanneke, 2006, Blum et al., 2004] for transductive learning from randomly selected $L$. These bounds are PAC style which typically show that, roughly, the error rate decreases with $O(\sum_{i,j} W_{i,j}|y_i - y_j|/(b|L|))$ where $b$ is the minimum 2-cut of the graph. Depending on the graph structure, our bounds can be significantly better. For example, if a binary weight graph contains $c$ cliques of size $n/c$ then, we can find an $\alpha$ cover of size $c\alpha \log(c\alpha)$ giving an error rate of $O(\sum_{i,j} W_{i,j}|y_i - y_j|/(n\alpha))$. This is better if $c \log(c\alpha) < n/b$.

A line of work has examined mistake bounds in terms of label smoothness for online learning on graphs [Pelckmans and Suykens, 2008, Brautbar, 2009, Herbster et al., 2008, 2005, Herbster, 2008]. These mistake bounds hold no matter how the sequence of vertices are chosen. Herbster [2008] also considers how cluster structure can improve mistake bounds in this setting and gives a mistake bound similar to our graph covering bound on prediction error. Herbster et al. [2005] discusses using an active learning method for the first several steps of an online algorithm. Our work differs from this previous work by considering prediction error bounds for offline learning as opposed to mistake bounds for online learning. The mistake bound setting is significantly different as the prediction method receives feedback after every prediction.

**Acknowledgments**

This material is based upon work supported by the National Science Foundation under grant IIS-0535100.

# References

P. Afshani, E. Chiniforooshan, R. Dorrigiv, A. Farzan, M. Mirzazadeh, N. Simjour, and H. Zarrabi-Zadeh. On the complexity of finding an unknown cut via vertex queries. In *COCOON*, 2007.

Y. Bengio, O. Delalleau, and N. Le Roux. Label propagation and quadratic criterion. In O. Chapelle, B. Schölkopf, and A. Zien, editors, *Semi-Supervised Learning*. MIT Press, 2006.

A. Blum and S. Chawla. Learning from labeled and unlabeled data using graph mincuts. In *ICML*, 2001.

A. Blum, J. Lafferty, M. R. Rwebangira, and R. Reddy. Semi-supervised learning using randomized mincuts. In *ICML*, 2004.

M. Brautbar. Online Learning a Labeling of a Graph. *Mining and Learning with Graphs*, 2009.

O. Chapelle, B. Schölkopf, and A. Zien. *Semi-supervised learning*. MIT press, 2006.

W. Cunningham. Optimal attack and reinforcement of a network. *Journal of the ACM*, 1985.

S. Fujishige. *Submodular Functions and Optimization*. Elsevier Science, 2005.

D. Gusfield. Very simple methods for all pairs network flow analysis. *SIAM Journal on Computing*, 1990.

S. Hanneke. An analysis of graph cut size for transductive learning. In *ICML*, 2006.

M. Herbster. Exploiting Cluster-Structure to Predict the Labeling of a Graph. In *ALT*, 2008.

M. Herbster, M. Pontil, and L. Wainer. Online learning over graphs. In *ICML*, 2005.

M. Herbster, G. Lever, and M. Pontil. Online Prediction on Large Diameter Graphs. In *NIPS*, 2008.

G. Karypis and V. Kumar. A fast and highly quality multilevel scheme for partitioning irregular graphs. *SIAM Journal on Scientific Computing*, 1999.

A. Krause, H. B. McMahan, C. Guestrin, and A. Gupta. Robust submodular observation selection. *JMLR*, 2008.

A. Y. Ng, M. I. Jordan, and Y. Weiss. On spectral clustering: Analysis and an algorithm. In *NIPS*, 2001.

K. Pelckmans and J. Suykens. An online algorithm for learning a labeling of a graph. In *Mining and Learning with Graphs*, 2008.

K. Pelckmans, J. Shawe-Taylor, J. Suykens, and B. De Moor. Margin based transductive graph cuts using linear programming. 2007.

A. Pucci, M. Gori, and M. Maggini. Semi-supervised active learning in graphical domains. In *Mining and Learning With Graphs*, 2007.

H. Saran and V. V. Vazirani. Finding k cuts within twice the optimal. *SIAM Journal on Computing*, 1995.

M. Wang, X. Hua, Y. Song, J. Tang, and L. Dai. Multi-Concept Multi-Modality Active Learning for Interactive Video Annotation. In *International Conference on Semantic Computing*, 2007.

W. Zhao, J. Long, E. Zhu, and Y. Liu. A scalable algorithm for graph-based active learning. In *Frontiers in Algorithmics*, 2008.

X. Zhu and J. Lafferty. Combining active learning and semi-supervised learning using gaussian fields and harmonic functions. In *ICML*, 2003.

